# Dyadic Classification Trees
# via
# Structural Risk Minimization

**Clayton Scott and Robert Nowak**
Department of Electrical and Computer Engineering
Rice University
Houston, TX 77005
{cscott,nowak}@rice.edu

## Abstract

Classification trees are one of the most popular types of classifiers, with ease of implementation and interpretation being among their attractive features. Despite the widespread use of classification trees, theoretical analysis of their performance is scarce. In this paper, we show that a new family of classification trees, called dyadic classification trees (DCTs), are near optimal (in a minimax sense) for a very broad range of classification problems. This demonstrates that other schemes (e.g., neural networks, support vector machines) cannot perform significantly better than DCTs in many cases. We also show that this near optimal performance is attained with linear (in the number of training data) complexity growing and pruning algorithms. Moreover, the performance of DCTs on benchmark datasets compares favorably to that of standard CART, which is generally more computationally intensive and which does not possess similar near optimality properties. Our analysis stems from theoretical results on structural risk minimization, on which the pruning rule for DCTs is based.

## 1 Introduction

Let $(X, Y) \in \mathbf{R}^d \times \{0, 1\}$ be a jointly distributed pair of random variables. In pattern recognition, $X$ is called an input vector, and contains the measurements from an experiment. The values in $X$ are referred to as features, attributes, or predictors. $Y$ is called a response variable, and is thought of as a class label associated with $X$. A classifier is a function $\phi : \mathbf{R}^d \to \{0, 1\}$ that attempts to match an input vector with the appropriate class. The performance of $\phi$ for a given distribution of the data is measured by the probability of error:

$$L(\phi) = \mathbf{P}\{\phi(X) \neq Y\}.$$

The classifier with the smallest probability of error, denoted $\phi^*$, is called the Bayes classifier. The Bayes classifier is given by

$$\phi^*(x) = \left\{ \begin{array}{ll} 1 & \text{if } \eta(x) > \frac{1}{2} \\ 0 & \text{otherwise} \end{array} \right. ,$$

where $\eta(x) = \mathbf{P}\{Y = 1 | X = x\} = \mathbf{E}\{Y|x\}$ is the regression of $Y$ on $X$. The probability of error for the Bayes classifier is denoted $L^*$.

The true distribution on the data is generally unknown. In such cases, we may construct a classifier $\phi_n$ based on a *training dataset* $D_n = \{(X_1, Y_1), \ldots, (X_n, Y_n)\}$ of independent, identically distributed samples. A procedure that constructs a classifier for all $n$ is called a *discrimination rule*. The performance of $\phi_n = \phi_n(x; D_n)$ is measured by

$$L_n = L(\phi_n) = \mathbf{P}\{\phi_n(X; D_n) \neq Y | D_n\},$$

the conditional probability of error. Note that $L_n$ is random, since $D_n$ is random.

In this paper, we examine a family of classifiers called *dyadic classification trees* (DCTs), built by recursive, dyadic partitioning of the input space. The appropriate tree from this family is obtained by building an initial tree (in a data-independent fashion), followed by a data-dependent pruning operation based on structural risk minimization (SRM). Thus, one important distinction between our approach and usual decision trees is that the initial tree is not adaptively grown to fit the data. The pruning strategy resembles that used by CART, except that the penalty assigned to a subtree is proportional to the square root of its size.

SRM penalized DCTs lead to a strongly consistent discrimination rule for input data $X$ with support in the unit cube $[0, 1]^d$. We also derive bounds on the rate of convergence of DCTs to the Bayes error. Under a modest regularity assumption (in terms of the box-counting dimension) on the underlying optimal Bayes decision boundary, we show that complexity-regularized DCTs converge to the Bayes decision at a rate of $n^{-1/(d+1)}$. Moreover, the minimax error rate for this class is at least $n^{-1/d}$. This shows that dyadic classification trees are near minimax-rate optimal, i.e., that no discrimination rule can perform significantly better in this minimax sense. We also present an efficient algorithm for implementing the pruning strategy, which leads to an $O(n)$ algorithm for DCT construction. The pruning algorithm requires $O(M \log M)$ operations to prune an initial tree with $M$ terminal nodes, and is based on the familiar pruning algorithm used by CART [1]. Finally, we compare DCTs with a CART-like tree classifier on four common datasets.

## 2   Dyadic Classification Trees

Throughout this work we assume that the input data is restricted to the unit hypercube, $[0, 1]^d$. This is a realistic assumption for real-world data, provided appropriate translation and scaling is applied. Let $\mathcal{P} = \{R_1, \ldots, R_k\}$ be a tree-structured partition of the input space, where each $R_i$ is a hyperrectangle with sides parallel to the coordinate axes. Given an integer $\ell$, let $[\ell]_d$ denote the element of $\{1, \ldots, d\}$ that is congruent to $\ell$ modulo $d$. If $R_i \in \mathcal{P}$ is a cell at depth $j$ in the tree, let $R_i^{(1)}$ and $R_i^{(2)}$ be the rectangles formed by splitting $R_i$ at its midpoint along coordinate $[j + 1]_d$. As a convention, assume $R_i^{(1)}$ contains those points of $R_i$ that are less than or equal to the midpoint along the dimension being split.

**Definition 1** *A* sequential dyadic partition *(SDP) is any partition of $[0, 1]^d$ that can be obtained by applying the following rules recursively:*

1. *The trivial partition $\mathcal{P} = \{[0, 1]^d\}$ is an SDP,*

2. *If $\mathcal{P} = \{R_1, \ldots, R_k\}$ is an SDP, then so is*

$$\{R_1, \ldots, R_{i-1}, R_i^{(1)}, R_i^{(2)}, R_{i+1}, \ldots, R_k\},$$

   *where $i$ may be any integer, $1 \leq i \leq k$.*

*We define a* dyadic classification tree *(DCT) to be a sequential dyadic partition with a class label (0 or 1) assigned to each node in the tree.*

The partitions are sequential because children must be split along the next coordinate after the coordinate where their parent was split. Such splits are referred to as *forced* splits, as opposed to *free* splits, in which any coordinate may be split. The partitions are dyadic because we only allow midpoint splits.

By a *complete* DCT of depth $L$, we mean a DCT such that every possible split up to depth $L$ has been made. In a complete DCT, every terminal node has volume $2^{-L}$. If $L$ is a multiple of $d$, then the terminal nodes of a complete DCT are hypercubes of sidelength $2^{-L/d}$.

## 3   SRM for DCTs

Structural risk minimization (SRM) is an inductive principle for selecting a classifier from a sequence of sets of classifiers based on complexity regularization. It was introduced by Vapnik and Chervonenkis (see [2]), and later analyzed by Lugosi and Zeger [3], [4, Ch. 18]. We formulate structural risk minimization for dyadic classification trees by applying results from [4, Ch. 18].

SRM is formulated in terms of the VC dimension, which we briefly review. Let $\mathcal{C}$ be a collection of classifiers $\phi : \mathbf{R}^d \rightarrow \{0, 1\}$, and let $z_1, \ldots, z_n \in \mathbf{R}^d$. If each of the $2^n$ possible labellings of $z_1, \ldots, z_n$ can be correctly classified by some $\phi \in \mathcal{C}$, we say $\mathcal{C}$ *shatters* $z_1, \ldots, z_n$. The *Vapnik-Chervonenkis dimension* (or VC dimension) of $\mathcal{C}$, denoted by $V$, is the largest integer $n$ for which there exist $z_1, \ldots, z_n \in \mathbf{R}^d$ such that $\mathcal{C}$ *shatters* $z_1, \ldots, z_n$. If $\mathcal{C}$ shatters some $n$ points for every $n$, then $V = \infty$ by definition. The VC dimension is a measure of the capacity of $\mathcal{C}$. As $V$ increases, $\mathcal{C}$ is able to separate more complex patterns.

If $m = 2^J$ for some integer $J \geq 0$, we say $m$ is *dyadic*. For dyadic $m$, and for $1 \leq k \leq m^d$, let $\mathcal{D}_m^{(k)}$ denote the collection of all DCTs with $k$ terminal nodes and depth not exceeding $dJ$, so that no terminal node has a side of length less than $1/m = 2^{-J}$. It is easily shown that the VC dimension of $\mathcal{D}_m^{(k)}$ is $k$ [5].

Given a dyadic integer $m$, and training data $\{(X_i, Y_i)\}_{i=1}^n$, for $k = 1, 2, \ldots, m^d$, define

$$\phi_{n,m}^{(k)} = \arg \min_{\phi \in \mathcal{D}_m^{(k)}} \widehat{L}_n(\phi),$$

where

$$\widehat{L}_n(\phi) = \frac{1}{n} \sum_{i=1}^n I(\phi(X_i) \neq Y_i)$$

is the empirical risk of $\phi$. Thus, $\phi_{n,m}^{(k)}$ is selected by empirical risk minimization over $\mathcal{D}_m^{(k)}$. Define the penalty term

$$\Delta(k, n) = \sqrt{\frac{32k \log(en)}{n}}, \tag{1}$$

and for $\phi \in \mathcal{D}_m^{(k)}$, define the penalized risk

$$\tilde{L}_n(\phi) = \widehat{L}_n(\phi) + \Delta(k, n).$$

The SRM principle selects the classifier $\phi_{n,m}^*$ from among $\phi_{n,m}^{(k)}, k = 1, 2, \ldots, m^d$, that minimizes $\tilde{L}_n(\phi_{n,m}^{(k)})$. We refer to $\phi_{n,m}^*$ as a *penalized* or *complexity-regularized dyadic classification tree*. We have the following risk bound.

**Theorem 1** *For all $n$ and $k \leq m^d$, and for all $\epsilon \geq 4\Delta(k, n)$,*

$$\mathbf{P} \left\{ L(\phi_{n,m}^*) - \inf_{\phi \in \mathcal{D}_m^{(k)}} L(\phi) > \epsilon \right\} \leq e^{-n\epsilon^2/128} + 8n^k e^{-n\epsilon^2/512},$$

*and in particular, for all $n$ and $m$,*

$$\mathbf{E}[L(\phi_{n,m}^*)] - L^* \leq \min_{1 \leq k \leq m^d} \left( 16\sqrt{\frac{k \log n + 4}{2n}} + \left( \inf_{\phi \in \mathcal{D}_m^{(k)}} L(\phi) - L^* \right) \right).$$

*Sketch of proof:* Apply Theorem 18.3 in [4] with $\mathcal{C}^{(k)} = \mathcal{D}_m^{(k)}$ and $V_k = k$ for $k = 1, 2, \ldots, m^d$. □

The first term on the right-hand side of the second bound is an upper bound on the expected estimation error. The second term is the approximation error. Even though the penalized DCT does not know the value of $k$ that optimally balances the two terms, it performs as though it does, because of the "min" in the expression. Nobel [6] gives similar results for classifiers based on initial trees that depend on the data.

The next result demonstrates strong consistency for the penalized DCT, where strong consistency means $L_n \to L^*$ with probabilty one.

**Theorem 2** *Suppose $n, m \to \infty$, with $m = m(n)$ assuming only dyadic integer values. If $m^d = o(n/\log n)$, then the penalized dyadic classification tree is strongly consistent for all distributions supported on the unit hypercube.*

*Sketch of proof:* The proof follows from the first part of Theorem 1 and strong universal consistency of the regular histogram classifier. See [5] for details. □

## 4   Rates of Convergence

In this section, we investigate bounds on the rate of convergence of complexity-regularized DCTs. First we obtain upper bounds on the rate of convergence for a particular class of distributions on $(X, Y)$. We then state a minimax lower bound on the rate of convergence of any data based classifier for this class.

Most rate of convergence studies in pattern recognition place a constraint on the regression function $\eta(x) = \mathbf{P}\{Y = 1 | X = x\}$ by requiring it to belong to a certain smoothness class (e.g. Lipschitz, Besov, bounded variation). In contrast, the class we study is defined in terms of the regularity of the Bayes decision boundary, denoted $\mathcal{B}$. We allow $\eta(x)$ to be arbitrarily irregular away from $\mathcal{B}$, so long as it is well behaved near $\mathcal{B}$. The Bayes decision boundary is informally defined as $\mathcal{B} = \{x : \eta(x) = 1/2\}$. A more rigorous definition should take into account the fact that $\eta$ might not take on the value $1/2$ [5].

We now define a class of distributions. Let $(X, Y)$ denote a random pair, as before, where $X$ takes on values in $[0, 1]^d$.

**Definition 2** *Let $C_1, C_2 > 0$. Define $\mathcal{F}(C_1, C_2)$ to be the collection of all distributions on $(X, Y)$ such that for all dyadic integers $m$, if we subdivide the unit cube into cubes of side length $1/m$,*

**A1** (Bounded marginal): *For any such cube $A$ intersecting the Bayes decision boundary, $\mathbf{P}\{X \in A\} \leq C_1 \lambda(A) = C_1/m^d$, where $\lambda$ denotes the Lebesgue measure.*

**A2** (Regularity): *The Bayes decision boundary passes through at most $C_2 m^{d-1}$ of the resulting $m^d$ cubes.*

*Define $\mathcal{F}$ to be the class of all $(X, Y)$ belonging to $\mathcal{F}(C_1, C_2)$ for some $C_1, C_2$.*

The first condition holds, for example, if the density of $X$ is essentially bounded with respect to the Lebesgue measure, with essential supremum $\leq C_1$. The second condition

can be shown to hold when one coordinate of the Bayes decision boundary is a Lipschitz function of the others. See, for example, the boundary fragment class of [7] with $\gamma = 1$ therein.

The regularity condition A2 is closely related to the notion of box-counting dimension of the Bayes decision boundary [8]. Roughly speaking, A2 holds for some $C_2$ if and only if the Bayes decision boundary has box-counting dimension $d - 1$. The box-counting dimension is an upper bound on the Hausdorff dimension, and the two dimensions are equal for most "reasonable" sets. For example, if $\mathcal{G}$ is a smooth $k$-dimensional submanifold of $\mathbf{R}^d$, then $\mathcal{G}$ has box-counting dimension $k$.

## 4.1 Upper Bounds on DCT Rate of Convergence

**Theorem 3** *Assume the distribution of $(X, Y)$ belongs to $\mathcal{F}(C_1, C_2)$. Let $\phi_{n,m}^*$ be the penalized dyadic classification tree, as described in Section 3. If $m \sim (n/\log n)^{1/(d+1)}$, then there exists a constant $C_3 > 0$ such that for all $n > 2$,*

$$\mathbf{E}[L(\phi_{n,m}^*)] - L^* \leq C_3 (\log n/n)^{1/(d+1)}.$$

When we write $m \sim (n/\log n)^{1/(d+1)}$, we mean $\log_2 m = \lfloor \log_2((n/\log n)^{1/(d+1)}) + C_0 \rfloor$, where $C_0 \in \mathbf{R}$ is arbitrary.

*Sketch of proof:* It can be shown that for each dyadic $m$, there exists a pruned DCT $\phi$ with $k = O(m^{d-1})$ leaf nodes, such that $L(\phi) - L^* \leq C_1 C_2/m$. Plugging this into the risk bound in Theorem 1 and minimizing over $m$ produces the desired result [5]. □

The minimal value of $C_3$ in the above theorem tends to $2C_1 C_2$ as $d \to \infty$. Note that similar rate of convergence results for data-grown trees would be more difficult to establish, since the approximation error is random in those cases.

It is possible to eliminate the log factor in the upper bound by means of Alexander's inequality, as discussed in [4, Ch. 12]. This leads to a much larger value of $C_3$, but an improved asymptotic rate.

To illustrate the significance of Theorem 3, consider a penalized histogram classifer, with bin width determined adaptively by structural risk minimization, as described in [4, Problem 18.6]. For that rule, the best exponent on the rate of convergence for our class is $1/(d + 2)$, compared with $1/(d + 1)$ for our rule. Intuitively, this is because the adaptive resolution of dyadic classification trees enables them to focus on the $d - 1$ dimensional decision boundary, rather than the $d$ dimensional regression function.

In the event that the data $X$ occupies a $d' < d$ dimensional subset of $[0, 1]^d$, the proof of Theorem 3 follows through as before, but with an exponent of $d' + 1$ instead of $d + 1$. Thus, the penalized DCT is able to automatically adapt to the dimensionality of the input data.

## 4.2 Minimax Lower Bound

The next result demonstrates that complexity-regularized DCTs nearly achieve the minimimax rate for our class of distributions.

**Theorem 4** *Let $\delta_n$ denote any discrimination rule based on training data. There exists a constant $C > 0$ such that for $n$ sufficiently large,*

$$\inf_{\delta_n} \sup_{\mathcal{F}} E[L(\delta_n)] - L^* \geq C n^{-1/d}.$$

*Sketch of proof:* This result follows from Theorem 2 in [7] (with $\gamma = \kappa = 1$ therein). The proof of that result is in turn based on Assouad's lemma. □

Theorems 3 and 4, together with the above remark on Alexander's inequality, show that complexity-regularized DCTs are close to minimax-rate optimal for the class $\mathcal{F}$. We suspect that the class studied by Tsybakov [7], used in our minimax proof, is more restrictive than our class. Therefore, it may be that the exponent $1/d$ in the above theorem can be decreased to $1/(d+1)$, in which case we achieve the minimax rate.

Although bounds on the minimax rate of convergence in pattern recognition have been investigated in previous work [9, 10], the focus has been on placing regularity assumptions on the regression function $\eta(x) = \mathbf{P}\{Y = 1|X = x\}$. Yang demonstrates that in such cases, for many common function spaces (e.g. Lipschitz, Besov, bounded variation), classification is not easier than regression function estimation [10]. This contrasts with the conventional wisdom that, in general, classification *is* easier than regression function estimation [4, Ch. 6]. Our approach is to study minimax rates for distributions defined in terms of the regularity of the Bayes decision boundary. With this framework, we see that minimax rates for classification can be orders of magnitude faster than for estimation of $\eta(x)$, since $\eta(x)$ may be arbitrarily irregular away from the decision boundary for distributions in our class. This view of minimax classification has also been adopted by Mammen and Tsybakov [7, 11]. Our contribution with respect to their work is an implementable discrimination rule, with guaranteed computational complexity, that nearly achieves the minimax lower bounds. We also remark that "fast rates" (e.g., $O(n^{-1})$) obtained by those authors require much stronger assumptions on the smoothness of the decision boundary and $\eta(x)$ than we employ in this paper.

## 5   An Efficient Pruning Algorithm

In this section we describe an algorithm to compute the penalized DCT efficiently. We switch notation, using $T$ to denote an arbitrary classification tree. Let $T' \leq T$ denote that $T'$ is a pruned version of $T$ (possibly $T$ itself). For $\alpha \in \mathbf{R}$, define

$$T_1(\alpha) = \arg\min_{T' \leq T} \ \widehat{L}_n(T') + \alpha|T'|,$$

and

$$T_2(\alpha) = \arg\min_{T' \leq T} \ \widehat{L}_n(T') + \alpha\sqrt{|T'|},$$

where $|T'|$ denotes the number of leaf nodes of $T'$. We are interested in computing $T_2(\alpha)$ when $T$ is a complete dyadic tree, and $\alpha = \sqrt{32\log(en)/n}$.

Breiman, et.al. [1] showed the existence of weights $-\infty = \alpha_0 < \alpha_1 < \cdots < \alpha_M = \infty$ and subtrees $T \geq S_1 \geq \cdots \geq S_M = \{root\}$ such that $T_1(\alpha) = S_i$ whenever $\alpha \in [\alpha_{i-1}, \alpha_i)$. Moreover, the weights $\alpha_i$ and subtrees $S_i$ may be found in $O(|T|\log|T|)$ operations [12, 13]. A similar result holds for the square-root penalty, and the trees produced are a subset of the trees produced by the additive penalty [5].

**Theorem 5** *For each $\alpha$, there exists $\alpha'$ such that $T_2(\alpha) = T_1(\alpha')$.*

Therefore, pruning $T$ with the square-root penalty always produces one of the trees $S_\ell$. We may then determine the pruned tree $T' \leq T$ minimizing the penalized risk $\widehat{L}_n(T') + \alpha\sqrt{|T'|}$ by minimizing this quantity over $S_i, i = 1, \ldots, M$. Thus, square-root pruning can be performed in $O(|T|\log|T|)$ operations.

In the context of constructing a penalized DCT, we start with an initial tree $T$ that is a complete DCT. For the classifiers in Theorems 2 and 3, this initial tree has size $|T| =$

Table 1: Comparison of a greedy tree growing procedure, with model selection based on holdout error estimate, and two DCT based methods. Numbers shown are test errors.

|  | CART-HOLD | DCT-HOLD | DCT-SRM |
|---|---|---|---|
| Pima Indian Diabetes | 26.8 % | 27.2 % | 33.0 % |
| Wisconsin Breast Cancer | 4.7 % | 6.4 % | 6.3 % |
| Ionosphere | 12.88 % | 18.6 % | 18.8 % |
| Waveform | 19.8 % | 29.1 % | 31.0 % |

$m^d = o(n/\log n)$, and so pruning requires $O(n)$ operations. Since the growing procedure also requires $O(n)$ operations, the overall construction is $O(n)$.

## 6  Experimental Comparison

To gain a rough idea of the usefulness of dyadic classification trees in practice, we compared two DCT based classifiers with a greedy tree growing procedure, similar to that used by CART [1] or C4.5 [14], where each successive split is chosen to maximize an information gain defined in terms of an impurity function. We considered four two-class datasets, available on the web at http://www.ics.uci.edu/~mlearn/MLRepository.html. For each dataset, we randomly split the data into two halves to form training and testing datasets.

For the greedy growing scheme, we used half of the training data to grow the tree, and constructed every possible pruning of the initial tree with an additive penalty. The best pruned tree was chosen to minimize the holdout error on the rest of the training data. We call this classifier CART-HOLD. The second classifier, DCT-HOLD, was constructed in a similar manner, except that the initial tree was a complete DCT, and all of the training data was used for computing the holdout error estimate. Finally, we implemented the complexity-regularized DCT, denoted DCT-SRM, with square-root penalty determined by Equation 1. Table 1 shows the misclassification rate for each algorithm on each dataset.

From these experiments, we might conclude two things: (i) The greedily-grown partition outperforms the dyadic partition; and (ii) Much of the discrepancy between CART-HOLD and DCT-SRM comes from the partitioning, and not from the model selection method (holdout versus SRM). Indeed, DCT-SRM beats or nearly equals DCT-HOLD on three of the four datasets. Conclusion (i) may be premature, for it is shown in [4, Ch. 20] that greedy partitioning based on impurity functions can perform arbitrarily poorly for some distributions, while this is never the case for complexity-regularized DCTs. In light of (ii), it may be possible to apply Nobel's pruning rules for data-grown trees [6], which can now be implemented with our algorithm, to equal or surpass the performance of CART, while avoiding the heuristic and computationally expensive cross-validation technique usually employed by CART to determine the appropriately pruned tree.

## 7  Conclusion

Dyadic classification trees exhibit desirable theoretical properties (finite sample risk bounds, consistency, near minimax-rate optimality) and can be trained extremely rapidly. The minimax result demonstrates that other discrimination rules, such as neural networks or support vector machines, cannot significantly outperform DCTs (in this minimax sense). This minimax result is asymptotic, and considers worst-case distributions. From a practical standpoint, with finite samples and non-worst-case distributions, other rules may beat DCTs, which our experiments on benchmark datasets confirm. The sequential dyadic partitioning scheme is especially susceptible when many of the features are irrelevant, since

it must cycle through all features before splitting a feature again. Several modifications to the current dyadic partitioning scheme may be envisioned, such as free dyadic or median splits.

Such modified tree induction strategies would still possess many of the desirable theoretical properties of DCTs. Indeed, Nobel has derived risk bounds and consistency results for classification trees grown according to data [6]. Our square-root pruning algorithm now provides a means of implementing his pruning schemes for comparison with other model selection techniques (e.g., holdout or cross-validation). It remains to be seen whether the rate of convergence analysis presented here extends to his work.

Further details on this work, including full proofs, may be found in [5].

## Acknowledgments

This work was partially supported by the National Science Foundation, grant no. MIP–9701692, the Army Research Office, grant no. DAAD19-99-1-0349, and the Office of Naval Research, grant no. N00014-00-1-0390.

## References

[1] L. Breiman, J. Friedman, R. Olshen, and C. Stone, *Classification and Regression Trees*, Wadsworth, Belmont, CA, 1984.

[2] V. Vapnik, *Estimation of Dependencies Based on Empirical Data*, Springer-Verlag, New York, 1982.

[3] G. Lugosi and K. Zeger, "Concept learning using complexity regularization," *IEEE Transactions on Information Theory*, vol. 42, no. 1, pp. 48–54, 1996.

[4] L. Devroye, L. Györfi, and G. Lugosi, *A Probabilistic Theory of Pattern Recognition*, Springer, New York, 1996.

[5] C. Scott and R. Nowak, "Complexity-regularized dyadic classification trees: Efficient pruning and rates of convergence," Tech. Rep. TREE0201, Rice University, 2002, available at http://www.dsp.rice.edu/~cscott.

[6] A. Nobel, "Analysis of a complexity based pruning scheme for classification trees," *IEEE Transactions on Information Theory*, vol. 48, no. 8, pp. 2362–2368, 2002.

[7] A. B. Tsybakov, "Optimal aggregation of classifiers in statistical learning," *preprint*, 2001, available at http://www.proba.jussieu.fr/mathdoc/preprints/.

[8] K. Falconer, *Fractal Geometry: Mathematical Foundations and Applications*, Wiley, West Sussex, England, 1990.

[9] J. S. Marron, "Optimal rates of convergence to Bayes risk in nonparametric discrimination," *Annals of Statistics*, vol. 11, no. 4, pp. 1142–1155, 1983.

[10] Y. Yang, "Minimax nonparametric classification–Part I: Rates of convergence," *IEEE Transactions on Information Theory*, vol. 45, no. 7, pp. 2271–2284, 1999.

[11] E. Mammen and A. B. Tsybakov, "Smooth discrimination analysis," *Annals of Statistics*, vol. 27, pp. 1808–1829, 1999.

[12] P. Chou, T. Lookabaugh, and R. Gray, "Optimal pruning with applications to tree-structured source coding and modeling," *IEEE Transactions on Information Theory*, vol. 35, no. 2, pp. 299–315, 1989.

[13] B. Ripley, *Pattern Recognition and Neural Networks*, Cambridge University Press, Cambridge, UK, 1996.

[14] R. Quinlan, *C4.5: Programs for Machine Learning*, Morgan Kaufmann, San Mateo, 1993.
